# ReLIZO: Sample Reusable Linear Interpolation-based Zeroth-order Optimization

**Xiaoxing Wang**[*], **Xiaohan Qin**[*], **Xiaokang Yang, Junchi Yan**[‡]
Dept. of CSE & School of AI & Moe Key Lab of AI, Shanghai Jiao Tong University
{figure1_wxx, galaxy-1, xkyang, yanjunchi}@sjtu.edu.cn

## Abstract

Gradient estimation is critical in zeroth-order optimization methods, which aims to obtain the descent direction by sampling update directions and querying function evaluations. Extensive research has been conducted including smoothing and linear interpolation. The former methods smooth the objective function, causing a biased gradient estimation, while the latter often enjoys more accurate estimates, at the cost of large amounts of samples and queries at each iteration to update variables. This paper resorts to the linear interpolation strategy and proposes to reduce the complexity of gradient estimation by reusing queries in the prior iterations while maintaining the sample size unchanged. Specifically, we model the gradient estimation as a quadratically constrained linear program problem and manage to derive the analytical solution. It innovatively decouples the required sample size from the variable dimension without extra conditions required, making it able to leverage the queries in the prior iterations. Moreover, part of the intermediate variables that contribute to the gradient estimation can be directly indexed, significantly reducing the computation complexity. Experiments on both simulation functions and real scenarios (black-box adversarial attacks neural architecture search, and parameter-efficient fine-tuning for large language models), show its efficacy and efficiency. Our code is available at `https://github.com/Thinklab-SJTU/ReLIZO.git`.

## 1 Introduction

Zeroth-order optimization (ZO) aims to estimate the gradients by only function evaluations and solve optimization problems by the descent method. It has been successfully applied to many fields, including science [41, 25], finance [35], and artificial intelligence [37, 45, 47], where the optimization functions are usually black-box without available gradients w.r.t. the variables. Typical ZO methods iteratively perform three major steps [35]: 1) estimate the gradient by sampling directions to update the variable and querying the function evaluation, 2) rectify the descent direction through momentum, element-wise sign operation, stochastic variance reduced algorithms, etc., and 3) update the variables according to the descent direction. Wherein, the first step, i.e. gradient estimation, is critical since it provides the essential direction to update variables, which have been explored by many recent works [5, 30].

Gradient estimation methods in ZO can be roughly categorized into two groups [5]: linear interpolation technique [4, 30] and smoothing strategy [39, 12]. The former proposes to sample a set of linearly independent directions to form an invertible matrix and compute the gradient through linear interpolation. Though such methods can get better descent directions, the sample size of directions should be the same as the dimension of variables, leading to an intractable complexity of building an invertible matrix and computing the matrix inversion as the dimension increases. A recent work [30]

makes the sample size in linear interpolation strategy independent of the dimension of variables yet involves an extra orthogonal requirement of the sampled vectors. The latter group proposes to estimate the gradient by a sum of directional derivatives along random directions e.g. Gaussian and uniform on a unit sphere. Such methods are simple to implement and the sample size directions are independent of the variable dimension. However, it provides a gradient estimation of a smoothed version of the objective function [35], which differs from the original one with a bound w.r.t. a smoothing parameter [39]. Additionally, the prior work [5] shows that smoothing methods require significantly more samples than linear interpolation strategies to obtain an accurate estimation The work [28] leverages information theory showing that under the Lipschitz continuous gradient assumption, ZO methods require sample complexities growing polynomially with the variable dimension.

Overall, the prior gradient estimation methods require large amounts of samples at each iteration to update variables. Nevertheless, in many real scenarios, one query of function evaluation can require large amounts of resources such as tasks in AutoML and reinforcement learning, which significantly restricts the acceptable sample size at each iteration and thus slow down the convergence of ZO methods. Consequently, it has been urgently demanded to reduce the number of queries while ensuring the convergence of ZO methods.

To this end, we delve into the ZO pipeline and observe that the magnitudes of steps to update variables are limited, making it possible to reuse the queries in the prior iterations. It is hoped that such a reusing strategy can be explored to significantly reduce the number of queries of function evaluation while maintaining the sample size unchanged at each iteration. However, it is hard to be directly applied to the prior gradient estimation methods. On one hand, for smoothing strategies, it is hard to guarantee the reused samples obey the demanded distribution, which will further increase the gradient estimation bias. On the other hand, methods based on linear interpolation techniques require a large computation complexity since building an invertible matrix needs to compute the null space of reused samples. Additionally, another work [30] requires orthogonal sampled vectors, which can hardly be satisfied by the prior samples, making it impossible to reuse the samples.

This work refers to Taylor's expansion and proposes to estimate gradients through a linear combination of limited direction samples and their queried function evaluations. We then model the gradient estimation as a quadratically constrained linear program (QCLP) and derive the analytical solution. Our method can reduce the required sample size (decoupled from the variable dimension) for linear interpolation strategies without the orthogonal condition required. Consequently, it supports to reuse the queries in the prior iterations. Moreover, part of the intermediate variables that contribute to the gradient estimation can be directly indexed from the computation in the prior iterations, significantly reducing the computation complexity during the ZO process. Theoretical analysis is performed to show that our method can be regarded as a general version of the traditional linear interpolation strategy and has a similar convergence rate. In summary, our contributions lie in:

**1) We introduce to reuse the prior queries to speed up the ZO procedure.** Unlike recent works that cannot reuse the queries due to the requirements of sampled vectors to be orthogonal in linear interpolation strategies or obey a specific distribution in smoothing techniques, we introduce to estimate the gradient with an arbitrary number of sampled vectors without orthogonal condition, making it possible to leverage the queries in the prior iterations. To the best of our knowledge, this work is the first that attempts to reuse queries to speed up the ZO procedure.

**2) We model the gradient estimation in ZO as a QCLP and derive the analytical solution by Lagrange multiplier.** Theoretical analysis also proves that our method has a similar convergence rate as conventional linear interpolation techniques. Moreover, after combining our gradient estimator based on QCLP with the reusing strategy, part of the intermediate variables that contribute to the analytical solution can also be directly indexed from the computation in the prior iterations, significantly reducing the complexity when the sample size is much lower than the variable dimension. Overall, our method is more efficient than linear interpolation-based methods with a comparable accurate estimation of the gradient.

**3) We conduct extensive experiments to show the efficacy and efficiency of our method.** We first compare our method with recent ZO methods on the CUTEst, showing that it has faster convergence speed and better solutions. Ablation studies show that the performance drop is negligible even with more than $50\%$ reuse rate. We then conduct experiments on the black-box adversarial attack task, showing that it has lower attack loss and $5\%$ better attack success rate compared to peer ZO solvers

with similar final $\ell_2$ distortion. Our method is further applied to the Neural Architecture Search (NAS) on NAS-Bench-201, outperforming other ZO solvers with $39\%$ less number of queries.

## 2 Related Work

**Zeroth-order optimization.** There are typically two types of gradient estimation in ZO algorithms: one-point estimation [38, 13, 54] and multi-point estimation [1, 39, 22, 19]. Since multi-point estimation is more accurate, some works [33, 52] made further research on ZO convex problems. As for the non-convex setting, the prior works [39, 22] proposed ZO-GD and its stochastic counterpart ZO-SGD. ZO algorithms have been widely employed in various scenarios, such as black-box adversarial attacks [11, 27], AutoML [47] and transfer learning [44]. It is worth noting that the gradient estimation is the most critical part of ZO solvers, which is the focus of the above methods.

**Gradient estimation via smoothing strategy.** Methods based on the smoothing strategy [3, 36] estimate gradients by averaging over several directional derivatives by sampling the direction from either a Gaussian distribution [39] or a uniform distribution [20] on a unit sphere. Referring to the prior smoothing strategy based ZO algorithms [39, 22], ZO-signSGD [34] yields faster convergence but only guarantees to converge to a neighborhood of a solution, and ZO-AdaMM [12] uses adaptive momentum to enhance the optimization.

**Gradient estimation via linear interpolation.** Methods based on linear interpolation [15, 4] estimate gradient by solving linear programming. Conn et al. [15] derived the error bounds between the true function value and interpolating polynomials, including linear and quadratic approximation models. Berahas et al.[4] showed that linear interpolation gives better gradients than Gaussian smoothing in derivative-free optimization. The prior work [5] derived bounds on the sample size and the sampling radius, showing that smoothing strategy requires significantly more samples to obtain an accurate estimation than linear interpolation. Kozak et al.[30] approximated the gradient by finite differences computed in a set of orthogonal random directions, which is a special case of linear interpolation. This work also focuses on the scope of linear interpolation due to its more accurate estimation of gradient than smoothing techniques.

**Remarks.** Our approach has few constraints on the sample size, similar to the smoothing techniques. Meanwhile, we estimate the gradients by solving a linear program problem, similar to the linear interpolation strategies. Overall, our approach enjoys the efficiency of smoothing techniques while maintaining estimation accuracy.

## 3 The Proposed ReLIZO Method

Similar to the prior ZO methods [22, 12], we consider the problem of finding a point $x^* \in \mathbb{R}^d$ such that $f(x^*) = \min_x f(x)$. The function $f : \mathbb{R}^d \to \mathbb{R}$ is black-box and satisfies the following assumptions:

**Assumption 3.1.** $0 < f(x^0) - f(x^*) < \infty$, where $x^0$ is the initial point.

**Assumption 3.2.** The function $f$ is continuously differentiable, and $\nabla f$ is $L$-Lipschitz continuous for $\forall x \in \mathbb{R}^d$.

### 3.1 Rethinking Gradient Estimation as QCLP

Consider the Taylor's expansion of function $f$:

$$f(y) = f(x) + \nabla f(x)^\top (y - x) + o(y - x). \tag{1}$$

The remainder can be omitted in many scenarios. For large-scale applications it can be hard to compute the number of function evaluations required to construct quadratic models.

Given $n$ random vectors as candidate updates to the variable $\Delta_x \in \mathbb{R}^{n \times d}$, we denote the i-th row of it as $\Delta_x^i$. The corresponding function evaluation $\{f(x + \Delta_x^i)\}_{i=1}^n$ can be queried and compose $\Delta_f \in \mathbb{R}^n$, whose entries $\Delta_f^i = f(x + \Delta_x^i) - f(x), \forall i = 1, ..., n$. According to Eq. 1, we can have the equations as follows:

$$\Delta_x \nabla f(x) \approx \Delta_f. \tag{2}$$

In this work, we mainly focus on the case where $n < d$, hoping to obtain approximately accurate gradients with fewer queries. To this end, we estimate the direction of gradient $\nabla f(x)$ as $g(x)$ by linearly combining the $n$ sampled vectors $\{\Delta_x^i\}_{i=1}^n$, yielding $g(x) = \Delta_x^\top \alpha$, where $\alpha \in \mathbb{R}^n$ represents the coefficients of the linear combination. The inner product of normalized $g(x)$ and $\nabla f(x)$ increases when their directions are getting closed, which can be formulated as:

$$\arg\max_g \frac{g(x)^\top \nabla f(x)}{\|g(x)\|} = \frac{\alpha^\top \Delta_x \nabla f(x)}{\sqrt{\alpha^\top \Delta_x \Delta_x^\top \alpha}}. \tag{3}$$

Substitute Eq. 2 into Eq. 3 and define $A_x = \Delta_x \Delta_x^\top \in \mathbb{R}^{n \times n}$, We can model the estimation of gradient direction as a quadratically constrained linear program (QCLP):

$$\arg\max_\alpha \ \alpha^\top \Delta_f, \qquad \text{s.t. } \alpha^\top A_x \alpha = 1. \tag{4}$$

We employ the Lagrange multiplier method to address the QCLP problem (4). The Lagrangian function is defined as: $L(\alpha, \lambda) = \alpha^\top \Delta_f - \lambda(\alpha^\top A_x \alpha - 1)$, where $\lambda$ is the Lagrange multiplier. Taking partial derivatives w.r.t. $\alpha$ and setting it to zero, we have $\frac{\partial L}{\partial \alpha} = \Delta_f - 2\lambda A_x \alpha = 0$ and $A_x \alpha = \frac{1}{2\lambda} \Delta_f$, yielding the direction of $\alpha \propto A_x^{-1} \Delta_f$. Consequently, we obtain the estimated direction of the gradient:

$$g(x) = \Delta_x^\top \alpha \propto \Delta_x^\top A_x^{-1} \Delta_f. \tag{5}$$

Finally, we initialize the magnitude of the descent direction $\sigma_k$ with learning rate $\eta$ and adopt the line search strategy to update $\sigma_k$ to satisfy the Armijo-Goldstein condition [2, 23].

## 3.2 Reusing the Queries in the Prior Iterations

Unlike recent works unable to reuse the queries due to the requirements of sampled vectors to be orthogonal in linear interpolation strategies [30] or to obey a specific distribution in smoothing techniques [39], our method has few constraints on the sampled vectors $\Delta_x$, making it possible to reuse the queries in the prior iterations.

Suppose $x_{k-1}$ is the variable and $\Delta_{x_{k-1}}$ is the $n$ random vectors at $(k-1)$-th iteration. We have the queried function values at points $\{x_{k-1} + \Delta_{x_{k-1}}^i\}_{i=1}^n$. The estimated descent direction is $g(x_{k-1})$ with step $\sigma_{k-1}$. After updating the variable at $(k-1)$-th iteration, we have $x_k = x_{k-1} - \sigma_{k-1} g(x_{k-1})$. Since $f(x_{k-1})$ has also been queried, we have $n+1$ queried points and can collect $n+1$ random vectors for $x_k$ with known queries at $k$-th iteration as:

$$\tilde{\Delta}_{x_k} = \{-\sigma_{k-1} g(x_{k-1})\} \cup \{-\sigma_{k-1} g(x_{k-1}) - \Delta_{x_{k-1}}^i\}_{i=1}^n. \tag{6}$$

Distances between $x_k$ and the prior $n+1$ queried points are $\|x_k - x_{k-1}\| = \|\sigma_{k-1} g(x_{k-1})\|$ and $\|x_k - (x_{k-1} + \Delta_{x_{k-1}}^i)\| = \|\sigma_{k-1} g(x_{k-1}) + \Delta_{x_{k-1}}^i\|, \forall i = \{1, ..., n\}$.

According to Taylor's expansion in Eq. 1, the remainder $o(y - x)$ will be non-negligible as $\|y - x\|$ increases. Therefore, we introduce a reusable distance bound $b$ to filter out the samples far away from the current point $x_k$, i.e., the reused vectors are obtained by:

$$\Delta_{x_k}^r = \{\Delta \in \tilde{\Delta}_{x_k} | \|\Delta\| < b\}. \tag{7}$$

After getting the reusable queries, we then randomly sample another $n - |\Delta_{x_k}^r|$ vectors $\{p_j\}$ to build $\Delta_{x_k} \in \mathbb{R}^{n \times d}$ for gradient estimation at $k$-th iteration:

$$\Delta_{x_k} = \Delta_{x_k}^r \cup \{p_j\}_{j=1}^{n - |\Delta_{x_k}^r|}. \tag{8}$$

Overall, in the $k$-th iteration, only $n - |\Delta_{x_k}^r|$ new queries should be conducted. The ablation study in Sec. 4.2 shows that as the sample size increases, even reusing 80% prior queries can also find good solutions.

**Discussion on the reusable distance bound $b$:** ZO methods estimate the directional derivative by difference and restrict the distances between the samples and current point to a small value, which has the same order as the stepsize (initial learning rate) $\eta \sim O(\frac{1}{d})$ as previous works (Table 1 in [29]). In our method, the reusable distance bound $b$ restricts the distances between the reusable samples and current point, which should has the same order of magnitude as the stepsize. Then we choose $b \sim O(\frac{1}{d})$. Ablation studies in Fig. 2 show that $b = 2\eta$ works pretty well in different optimization tasks with different sample sizes $N$, thus we set $b = 2\eta$ by default.

**Algorithm 1** ReLIZO

---

**Input:** A Black-box function $f(x)$; Sample size at each iteration $n$; Reusable distance bound $b$; Number of iterations $K$; Learning rate $\eta$.

**Output:** The best solution $x^*$ that minimizes $f(x)$.

1: $\tilde{\Delta}_{x_0} = \emptyset$;
2: **for** $k = 0$ to $K - 1$ **do**
3:     Collect the reusable vectors from the last iteration and get $\Delta_{x_k}^r$ by Eq. 6 and Eq. 7;
4:     Sample another $n - |\Delta_{x_k}^r|$ vectors $P$ and build $\Delta_{x_k} \in \mathbb{R}^{n \times d}$ by Eq. 8;
5:     Query the function evaluation for newly sampled vectors $f(x_k + p_j), \forall p_j \in P$;
6:     Compute $A_{x_k}$ by Eq. 11;
7:     Obtain the estimated gradient direction $g(x_k) \propto \Delta_{x_k}^\top A_{x_k}^{-1} \Delta_f$ by Eq. 5;
8:     Initialize the magnitude of descent direction $\sigma_k$ as $\eta$ and update $\sigma_k$ by the backtracking line search;
9:     Update the variable: $x_{k+1} \leftarrow x_k - \sigma_k g(x_k)$.
10: **end for**
11: Return the solution $x_K$ and the best solution $x^*$ satisfying $f(x^*) \geq f(x_k), \forall k \in \{0, ..., K\}$.

---

## 3.3 Reducing the Computation Complexity by Indexing

Eq. 5 shows that the estimated direction is related to the inverse of $A_x = \Delta_x \Delta_x^\top$. When the sample size is much smaller than the dimension of variables, i.e., $n << d$, the computation complexity of $A_x$ is $O(n^2 d)$ dominating the complexity of matrix inversion $O(n^3)$. Nevertheless, the complexity can be significantly reduced when a large number of rows of $\Delta_x$ is reused.

We denote $A_{x_{k-1}} = \Delta_{x_{k-1}} \Delta_{x_{k-1}}^\top$, which has been computed in the $(k-1)$-th iteration. In the $k$-th iteration, $\Delta_{x_k}^r$ in Eq. 7 is the reused samples. In the following, we introduce an efficient strategy to compute $\Delta_{x_k}^r \Delta_{x_k}^{r\top}$.

First, we define $\bar{\Delta}_{x_{k-1}} \in \mathbb{R}^{(n+1) \times d}$ to denote the vector differences between $x_{k-1}$ and all $n + 1$ reusable points including $\{x_{k-1} + \Delta_{x_{k-1}}^i\}_{i=1}^n \cup \{x_{k-1}\}$. Hence, we get $\bar{A}_{x_{k-1}} = \bar{\Delta}_{x_{k-1}} \bar{\Delta}_{x_{k-1}}^\top$ as:

$$\bar{\Delta}_{x_{k-1}} = \begin{bmatrix} \Delta_{x_{k-1}} \\ \hline \mathbf{0}^\top \end{bmatrix}, \quad \bar{A}_{x_{k-1}} = \begin{bmatrix} A_{x_{k-1}} & \vdots & \mathbf{0} \\ \hline \mathbf{0}^\top & \vdots & 0 \end{bmatrix}. \tag{9}$$

Without losing generality, we suppose $\Delta_{x_k}^r$ has $n_k$ rows and the $i$-th row of it reuses the $r_i$-th row vector in $\bar{\Delta}_{x_{k-1}}$, and $R = [r_1, ... r_{n_k}]$. Since $g(x_{k-1}) = \Delta_{x_{k-1}}^\top \alpha$, we have:

$$\Delta_{x_k}^r = -\underbrace{\sigma_{k-1} \alpha^\top \Delta_{x_{k-1}}}_{\mathbb{R}^{1 \times d}} - \underbrace{\bar{\Delta}_{x_{k-1}}^R}_{\mathbb{R}^{n_k \times d}}, \tag{10}$$

where $\bar{\Delta}_{x_{k-1}}^R = [\bar{\Delta}_{x_{k-1}}^{r_1}; ...; \bar{\Delta}_{x_{k-1}}^{r_{n_k}}] \in \mathbb{R}^{n_k \times d}$, and the subtraction should be broadcast among the first dimension. Then, we can derive $\Delta_{x_k}^r \Delta_{x_k}^{r\top}$ as follows:

$$\Delta_{x_k}^r \Delta_{x_k}^{r\top} = \underbrace{\sigma_{k-1}^2 \|g(x_{k-1})\|^2}_{\mathbb{R}^{1 \times 1}} + \underbrace{\bar{\Delta}_{x_{k-1}}^R \bar{\Delta}_{x_{k-1}}^{R\top}}_{\mathbb{R}^{n_k \times n_k}} - \underbrace{\sigma_{k-1} \bar{\Delta}_{x_{k-1}}^R \Delta_{x_{k-1}}^\top \alpha}_{\mathbb{R}^{n_k \times 1}} - \underbrace{\sigma_{k-1} [\bar{\Delta}_{x_{k-1}}^R \Delta_{x_{k-1}}^\top \alpha]^\top}_{\mathbb{R}^{1 \times n_k}},$$

where the addition and subtractions should broadcast among the corresponding dimensions. Note that $\bar{\Delta}_{x_{k-1}}^R \bar{\Delta}_{x_{k-1}}^{R\top}$ and $\bar{\Delta}_{x_{k-1}}^R \Delta_{x_{k-1}}^\top$ can be directly indexed from $\bar{A}_{x_{k-1}}$ (Eq. 9), that has been computed in the prior iterations. Hence, we can simplify Eq. 11 as:

$$\Delta_{x_k}^r \Delta_{x_k}^{r\top} = \sigma_{k-1}^2 \|g(x_{k-1})\|^2 + \bar{A}_{x_{k-1}}^{(R,R)} - \sigma_{k-1} \bar{A}_{x_{k-1}}^R \alpha - \sigma_{k-1} [\bar{A}_{x_{k-1}}^R \alpha]^\top,$$

where $R = [r_1, ... r_{n_k}]$ is the index of reusing vectors in $\bar{\Delta}_{x_{k-1}}$, $\bar{A}_{x_{k-1}}^{(R,R)} \in \mathbb{R}^{n_k \times n_k}$ denotes to index $R$ rows and $R$ columns from $\bar{A}_{x_{k-1}}$, $\bar{A}_{x_{k-1}}^R \in \mathbb{R}^{n_k \times n}$ denotes to index $R$ rows from $\bar{A}_{x_{k-1}}$, and $\alpha \in \mathbb{R}^{n \times 1}$ is the solution to QCLP problem (4). Hence, by reusing $n_k$ vectors $\Delta_{x_k}^r$ and newly sampled vectors $P \in \mathbb{R}^{(n-n_k) \times d}$, $A_{x_k} = \Delta_{x_k} \Delta_{x_k}^\top$ can be obtained by:

$$A_{x_k} = \begin{bmatrix} \Delta_{x_k}^r \Delta_{x_k}^{r\top} & \vdots & \Delta_{x_k}^r P^\top \\ \hline (\Delta_{x_k}^r P^\top)^\top & \vdots & P P^\top \end{bmatrix}. \tag{11}$$

Consequently, the complexity of computing $A_{x_k}$ can be reduced to $O(n_k n + (n - n_k)nd)$, which decreases as the reuse rate $\frac{n_k}{n}$ increases. The overall computation complexity is analyzed and compared with other ZO methods in Appendix B. The algorithm flow is shown in Alg. 1.

## 3.4 Convergence Analysis

We start the analysis with a lemma that estimates the distance between the estimated gradient in our method and the projected exact gradient of the function.

**Lemma 3.1.** *Let $f$ satisfies Assumption 3.1 and 3.2. For every $k \in \mathbb{N}$, $\Delta_{x_k} \in \mathbb{R}^{n \times d}$ and $\|\Delta_{x_k}^i\| < b$:*

$$\|\Delta_{x_k} g(x_k) - \Delta_{x_k} \nabla f(x_k)\| \leq \frac{Lb\sqrt{n}}{2}. \tag{12}$$

Then we can derive the distance bound between the estimated gradient and the exact gradient of the function.

**Lemma 3.2.** *Let the SVD of $\Delta_{x_k} = U\Lambda V^\top$, $A_{x_k}^{-1} = (\Delta_{x_k}\Delta_{x_k}^\top)^{-1}$, $\rho_k$ denotes the spectral radius of $A_{x_k}^{-1}$. Based on Lemma 3.1, we have:*

$$\|g(x_k) - \nabla f(x_k)\|^2 \leq r\|\nabla f(x_k)\|^2 + \frac{\rho_k L^2 b^2 n}{4}, \tag{13}$$

*where $r = \frac{\sum_{j=n}^d v_j^2}{\|\nabla f(x_k)\|^2} < 1, \quad v = V^\top \nabla f(x_k)$.*

Consequently, we can derive the following proposition:

**Proposition 3.3.** *Suppose $f$ is convex and $\exists\, \theta \in [0, 1)$ satisfying $\rho_k \leq \frac{4(\theta^2 - r)\|\nabla f(x)\|^2}{L^2 b^2 n}$, we have:*

$$f(x_k) - f(x^*) \leq \frac{C}{k(1-\theta)^2} \sim O(\frac{d}{k}), \tag{14}$$

which yields a sublinear convergence. Proof can be found in Appendix A. Furthermore, our experimental results demonstrate that our method can converge in many scenarios, including simulation benchmarks and real-world applications.

## 3.5 Relationship with Linear Interpolation Methods

We show that our method can be regarded as a general version of the linear interpolation strategy.

First, consider the situation where $n = d$ and the matrix $\Delta_x$ is nonsingular, the estimated gradient by our method in Eq. 5 and be simplified as $g(x) \propto \Delta_x^\top A_x^{-1} \Delta_f = \Delta_x^{-1} \Delta_f$, which has a similar formulation to the gradient estimated by linear interpolation strategy [5].

Second, consider a more general situation where $n < d$, the matrix $\Delta_x$ can be singular, and its singular value decomposition (SVD) is denoted as $\Delta_x = U\Lambda V^\top$. Then $A_x = \Delta_x \Delta_x^\top = U\Lambda\Lambda^\top U^\top$, and the estimated gradient by our method in Eq. 5 can be transformed as:

$$g(x) \propto \Delta_x^\top A_x^{-1} \Delta_f = V\Lambda^\top U^\top (U\Lambda\Lambda^\top U^\top)^{-1}\Delta_f = V\Lambda^\top (\Lambda\Lambda^\top)^{-1} U^\top \Delta_f. \tag{15}$$

Since $U$ and $V$ are orthonormal, we have $\Delta_x(V\Lambda^\top(\Lambda\Lambda^\top)^{-1}U^\top)\Delta_x = \Delta_x$, showing that $V\Lambda^\top(\Lambda\Lambda^\top)^{-1}U^\top$ is the pseudo-inverse of $\Delta_x$. Consequently, our method can be regarded as a general version of the linear interpolation strategy.

Moreover, in the spacial case where $\Delta_x$ is orthonormal, $\Lambda\Lambda^\top = \mathbb{E}$ is an identity matrix, and Eq. 15 can be simplified as $g(x) \propto V\Lambda^\top U^{-1}\Delta_f = \Delta_x^\top \Delta_f$, consistent with the gradient estimation in [30].

# 4 Experiment

## 4.1 Protocols

To demonstrate the efficacy and efficiency of our method, we conduct experiments on both simulation benchmarks and real-world scenarios. First, we test on the CUTEst [24], one of the most popular

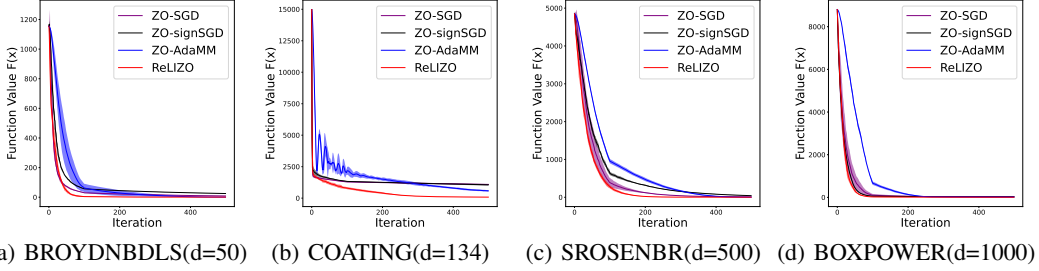

(a) BROYDNBDLS(d=50)    (b) COATING(d=134)    (c) SROSENBR(d=500)    (d) BOXPOWER(d=1000)

Figure 1: Illustration of the optimization procedure of different solvers on four problems. Each solver conducts 500 iterations to update the variables. The dimension of variable $d$ is shown in the bracket in the title of each figure. We run each solver at each setting three times and plot the average performance and standard deviation as the line and shadow.

benchmarks for derivative-free optimization containing over 1,500 problems, many of which are parametrized to allow for variable dimensions through user-selectable parameters. Specifically, we utilize the PyCUTEst [21], a Python interface, to import the optimization problems from the CUTEst and solve them by multiple ZO solvers implemented ourselves on the PyTorch platform. Second, we conduct experiments on the black-box adversarial attack task, a real-world application for ZO solvers. The task attempts to search additive noises for each pixel of inputs to confuse a pretrained neural network, which is one of the most popular scenarios for ZO solvers. Third, we apply our method to the Neural Architecture Search (NAS) task, aiming to search for the best neural architecture. Since querying once in NAS task consumes large amounts of resources, we conduct experiments on NAS-Bench-201 [18], which can directly query the performance of 15,625 architectures on three datasets including CIFAR10, CIFAR100, and ImageNet-16-120. Finally, we conduct experiment on large-scale neural network training by fine-tuning large language models (LLMs), demonstrating the applicability of our proposed ReLIZO.

## 4.2 Experiments on the Simulation Benchmark

We compare the performance of our method with baselines on four problems from the CUTEst optimization benchmark, containing an ill-conditioned or singular type problem (BOXPOWER), a nonlinear optimization test problem (SROSENBR), and a problem coming from application scenarios (COATING). The dimension of variables ranges from 50 to 1000 to evaluate the effectiveness of our method on both low-dimension and high-dimension problems. The baselines include ZO-SGD [22], ZO-signSGD [34] and ZO-AdaMM [12] implemented by ourselves on the PyTorch platform. Each solver updates the variables 500 times and samples 8 random directions at each iteration to update the variables. We also utilize grid search to obtain the best learning rate for each problem. The candidate learning rate $\eta$ ranges from {0.0001, 0.0002, 0.0005, 0.001, 0.002, 0.005, 0.01, 0.02, 0.05, 0.1, 0.2, 0.5}. As for our method, the total sample size at each iteration is set as 8, and the reusable distance bound b is set as $2\eta$, where $\eta$ is the learning rate obtained by the grid search. The variables are initialized as the default values in CUTEst for all solvers. Experiments at each setting are repeated three times, and the results are shown in Fig. 1. We observe that our method has a faster convergence speed and achieves better solutions to different problems. Moreover, the standard deviation of our method among three replications (shadows in Fig. 1) is much lower than other methods, showing that our method can stably find good solutions.

**Ablation Study about sample size $N$ and reusable distance bound $b$.** To evaluate the impact of sample size at each iteration and reusable distance bound, we run our method for 500 iterations on different sample sizes ranging from {6, 8, 10, 20, 50, 100, 200} and reusable distance bounds ranging from {0, $\eta$, $2\eta$, $5\eta$}, where $\eta$ is the learning rate at each iteration. Note that we set the sample size smaller than the dimension of variables. Results are shown in Fig. 2. The x-axis indicates the sample size, the y-axis shows the optimal value found by our solver after 500 iterations, and colors indicate different reusable distance bounds. For each setting, we conduct three replication experiments and plot the average performance and standard deviation as the nodes and lines. We also illustrate the reusing rates $= \frac{\text{\# reused queries}}{\text{total sample size}}$ in Fig. 2 as the floats in the boxes beside the scatters. According to the

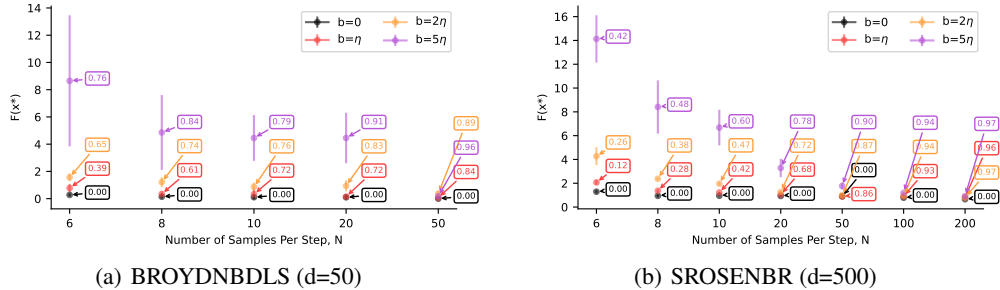

(a) BROYDNBDLS (d=50)  (b) SROSENBR (d=500)

Figure 2: Illustration of the best function value found by our method in 500 iterations under different settings of sample size $N$ and reusable distance bound $b$. We conduct three replication experiments at each setting and plot the average performance and standard deviation as nodes and lines. We also illustrate the reusing rates as the floats in the boxes beside the scatters.

results, we have three observations: **1)** A larger reusable distance bound $b$ has a larger reusable rate, and a larger sample size will also lead to a larger reusable rate. **2)** As the sample size increases, large reusable distance bound can also find good solutions, demonstrating the rationality of reusing queries. **3)** the standard deviation of performance is pretty small when $N > 6$ and $b < 5\eta$, showing that it can stably find good solutions. Experiments and analysis on extreme conditions (e.g. small $N$ and large $d$) in Appendix C.1 also demonstrate the robustness and scalability of ReLIZO.

## 4.3 Experiments on Black-Box Adversarial Attacks

ZO methods are popular solutions to black-box adversarial attacks, and we conduct experiments to show the effectiveness of our ReLIZO. We focus on universal adversarial perturbation against images [27, 14] and attack a well-trained DNN model on the CIFAR-10 dataset. Supported by [8] and [12], we consider the optimization problem as: $\arg\min_{\boldsymbol{\delta}} \quad \frac{\lambda}{M}\sum_{i=1}^{M} f(\boldsymbol{x}_i + \boldsymbol{\delta}) + \|\boldsymbol{\delta}\|_2^2$ s.t. $(\boldsymbol{x}_i + \boldsymbol{\delta}) \in [-0.5, 0.5]^d, \forall i$, where $f(\boldsymbol{x}_i + \boldsymbol{\delta})$ denotes the black-box attack loss for image $i$, $\lambda > 0$ is a regularization hyperparameter between minimizing the attack loss and the $\ell_2$ distortion. We normalize the pixel values to $[-0.5, 0.5]^d$, and we specify the loss function for untargeted attack as $f(x') = \max_{j \neq t}\{Z(x')_t - \max_{j \neq t} Z(x')_j - \kappa\}$, where $Z(x')_k$ denotes the prediction score of class $k$ given the input $x'$, and $\kappa > 0$ governs the gap between

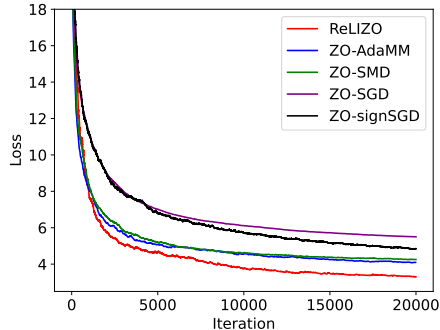

Figure 3: Attack loss of universal attack.

the confidence of the predicted label and the true label $t$. In experiments, we set $\kappa = 0$, and the attack loss $f$ reaches the minimum value 0 as the perturbation succeeds to fool the network.

We conduct experiments on the CIFAR-10 test dataset. Similar to the settings in ZO-AdaMM [12], we randomly select 100 images, ensuring that these images are initially correctly classified by the model. We conduct adversarial attacks on these selected images and compare the performance of our proposed ReLIZO method with 4 existing methods: ZO-SGD [22], ZO-signSGD [34], ZO-SMD [19], and ZO-AdaMM [12]. We conduct a linear search for the learning rate of each approach to maintain consistency in the $\ell_2$ distortion when employing each method for the at-

Table 1: Summary of attack success rate using different zeroth-order optimization methods with the final $\ell_2$ distortion remains relatively consistent. The result is based on a universal attack against 100 images under $T = 20000$ iterations.

| Methods | Attack success rate | Final $\ell_2$ distortion |
|---|---|---|
| ZO-SGD | 77% | 1.34 |
| ZO-signSGD | 80% | 1.36 |
| ZO-SMD | 85% | 1.38 |
| ZO-AdaMM | 87% | 1.37 |
| **ReLIZO** | **92%** | 1.37 |

Table 2: Top-1 test classification accuracy (%) on NAS-Bench-201. The first block shows the performance of gradient-based methods quoted from the paper of NAS-Bench-201. The second block shows the performance of various ZO methods, which are implemented by ourselves on the PyTorch platform. The performance of the methods based on ZO optimizers is averaged over three independent trials.

| Methods | CIFAR-10 | | CIFAR-100 | | ImageNet-16-120 | |
|---|---|---|---|---|---|---|
| | valid | test | valid | test | valid | test |
| ENAS [40] | 37.51±3.19 | 53.89±0.58 | 13.37±2.35 | 13.96±2.33 | 15.06±1.95 | 14.84±2.10 |
| RSPS [31] | 80.42±3.58 | 84.07±3.61 | 52.12±5.55 | 52.31±5.77 | 27.22±3.24 | 26.28±3.09 |
| DARTS-V1 [32] | 39.77±0.00 | 54.30±0.00 | 15.03±0.00 | 15.61±0.00 | 16.43±0.00 | 16.32±0.00 |
| DARTS-V2 [32] | 39.77±0.00 | 54.30±0.00 | 15.03±0.00 | 15.61±0.00 | 16.43±0.00 | 16.32±0.00 |
| GDAS [17] | 89.89±0.08 | 93.61±0.09 | 71.34±0.04 | 70.70±0.30 | 41.59±1.33 | 41.71±0.98 |
| SETN [16] | 84.04±0.28 | 87.64±0.00 | 58.86±0.06 | 59.05±0.24 | 33.06±0.02 | 32.52±0.21 |
| ZO-AdaMM [12] | 85.97±2.28 | 89.59±2.25 | 63.24±4.68 | 63.28±4.86 | 35.43±4.07 | 35.63±4.73 |
| ZO-SGD [22] | 88.18±1.26 | 91.60±1.15 | 67.59±1.51 | 67.49±1.05 | 39.83±1.80 | 39.79±1.83 |
| ZO-signSGD [34] | 85.94±4.37 | 89.74±3.64 | 61.89±8.82 | 62.41±8.91 | 35.23±6.82 | 35.28±7.51 |
| **ReLIZO (ours)** | 89.50±1.53 | 92.45±1.84 | 69.00±2.27 | 69.03±1.98 | 42.09±3.55 | 42.31±3.87 |

tacks, and compared the loss and the attack success rate on this basis. Fig. 3 illustrates the overall loss curves, while Table 1 presents the success rates of the adversarial attacks for each method over $T = 20000$ iterations. As we can see, our ReLIZO method exhibits the fastest rate of loss reduction compared to other ZO methods. Simultaneously, it achieves the highest attack success rate (more than 5% improvement) under condition where the $\ell_2$ distortion remains relatively consistent. This result fully demonstrates the effectiveness of our method.

## 4.4 Experiments on Neural Architecture Search

AutoML, including neural architecture search [32, 40, 51, 50, 49, 46, 53] and hyper-parameter optimization tasks [7, 43, 26, 48], has attracted wide attention due to its efficacy in reducing labor costs. Zero-order optimization is an effective algorithm for solving AutoML tasks [47], which can be modeled as bi-level optimization.

Table 3: Number of queries and the search cost (seconds) during the NAS procedure on the search space of NAS-Bench-201.

| Methods | ZO-AdaMM | ZO-SGD | ZO-signSGD | **ReLIZO** |
|---|---|---|---|---|
| **#Queries** | 19550 | 19550 | 19550 | 11885 |
| **Cost (s)** | 63044 | 63429 | 69659 | 55005 |

Table 2 reports the performance of our method on the NAS-Bench-201, a popular benchmark in the NAS task [32, 17, 51]. The first block shows the performance of gradient-based methods which is obtained from the paper [18]. The second block shows the performance of various ZO methods, including Zo-AdaMM, ZO-SGD, ZO-signSGD and ours. Specifically, similar to DARTS, we iteratively train the operation weights by the SGD optimizer and train the architecture parameters by the ZO optimizer for 50 epochs on the CIFAR10 dataset. To query the performance of one sampled direction, we update the architecture parameters by the direction and return the loss after training the operation weights for 10 iterations. The accuracy of discovered architectures on three datasets is directly indexed from the benchmark. For each ZO optimizer, we conduct three replication experiments and report the averaged accuracy and standard deviation in Table 2. We observe that our method surpasses other ZO methods in performance. We also report the number of queries of each ZO optimizer during the whole search process and the search time in Table 3. Our method is much faster than other ZO optimizers with fewer queries. Notice that the queries in Table 3 are used to train the architecture parameters, while the cost consists of training operation weights and architecture parameters, making the reduction of cost lower than the reduction of queries.

## 4.5 Experiments on Parameter Efficient Fine-tuning

Compared to gradient-based methods, ZO methods are well-suited for black-box optimization problems where gradients with respect to the variables are unavailable. Additionally, ZO methods are more memory-efficient than gradient-based methods since they do not require a backward pass,

Table 4: Zero-order optimizers with different parameter efficient fine-tuning methods on OPT-1.3b model (with 1.3 billion parameters) on the Stanford Sentiment Treebank v2 (SST2) task.

| Methods | Fine-Tuning | | LoRA | | Prefix-Tuning | | Prompt-Tuning | |
|---|---|---|---|---|---|---|---|---|
| | Acc | Memory | Acc | Memory | Acc | Memory | Acc | Memory |
| SGD | 91.1 | 44.1 GB | 93.6 | 44.1 GB | 93.1 | 42.9 GB | 92.8 | 50.6 GB |
| ZO-SGD | 90.8 | 28.7 GB | 90.1 | 19.7 GB | 91.4 | 20.7 GB | 84.4 | 18.8 GB |
| ZO-SGD-Sign | 87.2 | 31.4 GB | 91.5 | 19.7 GB | 89.5 | 20.7 GB | 72.9 | 18.8 GB |
| ZO-Adam | 84.4 | 31.4 GB | 92.3 | 19.7 GB | 91.4 | 20.7 GB | 75.7 | 18.8 GB |
| **ReLIZO (ours)** | **93.4** | 35.7 GB | **93.1** | 19.9 GB | **91.8** | 20.7 GB | **90.1** | 18.9 GB |

making them applicable to network training, as demonstrated by recent works [10, 55]. Specifically, DeepZero [10] introduces a principled ZO framework for deep learning that is computational-graph-free and can scale to deep neural network training with performance comparable to first-order methods. The work [55] also applies ZO methods to large language model (LLM) fine-tuning, highlighting their memory efficiency and introducing a ZO-LLM benchmark.

To illustrate the applicability of ReLIZO in network training, we adopt it for fine-tuning an OPT-1.3b model (with 1.3 billion parameters) on the Stanford Sentiment Treebank v2 (SST2) task with a batch size 128, following the methodology of ZO-LLM [55]. The results, shown in Table 4 indicate that ReLIZO outperforms other ZO methods across various fine-tuning schemes, including full parameter fine-tuning (FT), LoRA, Prefix-tuning, and Prompt-tuning. Notably, ReLIZO even surpasses SGD in the FT scheme while requiring significantly less memory, demonstrating its promising potential.

## 5   Conclusion, Limitations and Future Work

This work handles gradient estimation in zeroth-order optimization by modeling it as a quadratically constrained linear program problem, with analytical solution derived via the Lagrange multiplier method. Our method ReLIZO, decouples the required sample size from the variable dimension without the requirement of orthogonal condition in the recent linear interpolation works, making it possible to leverage the queries in the prior iterations. We also show that part of the intermediate variables contributing to the gradient estimation can be directly indexed, significantly reducing the computation complexity. We further perform a theoretical analysis of the convergence of ReLIZO and show the relationship with the prior linear interpolation methods. The efficacy and efficiency have been shown in extensive experiments, including simulated benchmarks, NAS, and black-box adversarial attack tasks.

**Limitations and Future work.** Compared to the gradient-based methods, ZO methods require multiple queries of function evaluation at each iteration, which restricts their applications in real-world scenarios such as neural network training. Our ReLIZO can reduce the required number of queries by the reusing strategy, potentially broadening the scope of applications of ZO, which can be explored in future work. However, in cases where the consumption of one query is negligible, the profit due to the reusing technique decreases, causing a relative increase in the cost of solving the QCLP. We leave these problems in future work.

## Impact Statements

This paper presents work whose goal is to advance the field of Machine Learning. Note that zeroth-order optimization is an effective method for black-box adversarial attacks as we have shown in Section 4.3. Its purpose is to evaluate the robustness of machine learning models, but we should also be aware of the impact this may have on AI safety. There are many potential societal consequences of our work, yet none of which we think must be specifically highlighted here.

## Footnotes

* Equal contribution; ‡ Corresponding author. This work was partly supported by NSFC (92370201, 62222607) and Shanghai Municipal Science and Technology Major Project under Grant 2021SHZDZX0102.

# References

[1] Alekh Agarwal, Ofer Dekel, and Lin Xiao. Optimal algorithms for online convex optimization with multi-point bandit feedback. In *Colt*, pages 28–40. Citeseer, 2010.

[2] Larry Armijo. Minimization of functions having lipschitz continuous first partial derivatives. *Pacific Journal of mathematics*, 16(1):1–3, 1966.

[3] Anastasia Bayandina, Alexander Gasnikov, Fariman Guliev, and Anastasia Lagunovskaya. Gradient-free two-points optimal method for non smooth stochastic convex optimization problem with additional small noise. *arXiv preprint arXiv:1701.03821*, 2017.

[4] Albert S Berahas, Liyuan Cao, Krzysztof Choromanski, and Katya Scheinberg. Linear interpolation gives better gradients than gaussian smoothing in derivative-free optimization. *arXiv preprint arXiv:1905.13043*, 2019.

[5] Albert S. Berahas, Liyuan Cao, Krzysztof Choromanski, and Katya Scheinberg. A theoretical and empirical comparison of gradient approximations in derivative-free optimization. *Found. Comput. Math.*, 22(2):507–560, 2022.

[6] Albert S Berahas, Liyuan Cao, and Katya Scheinberg. Global convergence rate analysis of a generic line search algorithm with noise. *SIAM Journal on Optimization*, 31(2):1489–1518, 2021.

[7] James Bergstra and Yoshua Bengio. Random search for hyper-parameter optimization. *Journal Of Machine Learning Research*, 2012.

[8] Nicholas Carlini and David Wagner. Towards evaluating the robustness of neural networks. In *2017 ieee symposium on security and privacy (sp)*, pages 39–57. Ieee, 2017.

[9] Richard G Carter. On the global convergence of trust region algorithms using inexact gradient information. *SIAM Journal on Numerical Analysis*, 28(1):251–265, 1991.

[10] Aochuan Chen, Yimeng Zhang, Jinghan Jia, James Diffenderfer, Konstantinos Parasyris, Jiancheng Liu, Yihua Zhang, Zheng Zhang, Bhavya Kailkhura, and Sijia Liu. Deepzero: Scaling up zeroth-order optimization for deep model training. In *ICLR*, 2024.

[11] Pin-Yu Chen, Huan Zhang, Yash Sharma, Jinfeng Yi, and Cho-Jui Hsieh. Zoo: Zeroth order optimization based black-box attacks to deep neural networks without training substitute models. In *AISEC*, pages 15–26, 2017.

[12] Xiangyi Chen, Sijia Liu, Kaidi Xu, Xingguo Li, Xue Lin, Mingyi Hong, and David Cox. Zo-adamm: Zeroth-order adaptive momentum method for black-box optimization. *NeurIPS*, 32, 2019.

[13] Xin Chen, Yujie Tang, and Na Li. Improve single-point zeroth-order optimization using high-pass and low-pass filters. In *ICML*, volume 162, pages 3603–3620, 2022.

[14] Minhao Cheng, Thong Le, Pin-Yu Chen, Jinfeng Yi, Huan Zhang, and Cho-Jui Hsieh. Query-efficient hard-label black-box attack: An optimization-based approach. *arXiv preprint arXiv:1807.04457*, 2018.

[15] Andrew R Conn, Katya Scheinberg, and Luís N Vicente. Geometry of interpolation sets in derivative free optimization. *Mathematical programming*, 111:141–172, 2008.

[16] Xuanyi Dong and Yi Yang. One-shot neural architecture search via self-evaluated template network. In *ICCV*, pages 3680–3689. IEEE, 2019.

[17] Xuanyi Dong and Yi Yang. Searching for a robust neural architecture in four GPU hours. In *CVPR*, pages 1761–1770. Computer Vision Foundation / IEEE, 2019.

[18] Xuanyi Dong and Yi Yang. Nas-bench-201: Extending the scope of reproducible neural architecture search. In *ICLR*. OpenReview.net, 2020.

[19] John C Duchi, Michael I Jordan, Martin J Wainwright, and Andre Wibisono. Optimal rates for zero-order convex optimization: The power of two function evaluations. *IEEE Transactions on Information Theory*, 61(5):2788–2806, 2015.

[20] Abraham D Flaxman, Adam Tauman Kalai, and H Brendan McMahan. Online convex optimization in the bandit setting: gradient descent without a gradient. *arXiv preprint cs/0408007*, 2004.

[21] Jaroslav Fowkes, Lindon Roberts, and Árpád Bürmen. Pycutest: an open source python package of optimization test problems. *J. Open Source Softw.*, 7(78):4377, 2022.

[22] Saeed Ghadimi and Guanghui Lan. Stochastic first-and zeroth-order methods for nonconvex stochastic programming. *SIAM Journal on Optimization*, 23(4):2341–2368, 2013.

[23] Allen A Goldstein. On steepest descent. *Journal of the Society for Industrial and Applied Mathematics, Series A: Control*, 3(1):147–151, 1965.

[24] Nicholas I. M. Gould, Dominique Orban, and Philippe L. Toint. Cutest: a constrained and unconstrained testing environment with safe threads for mathematical optimization. *Comput. Optim. Appl.*, 60(3):545–557, 2015.

[25] Genetha Anne Gray, Tamara G. Kolda, Ken Sale, and Malin M. Young. Optimizing an empirical scoring function for transmembrane protein structure determination. *INFORMS J. Comput.*, 16(4):406–418, 2004.

[26] Frank Hutter, Holger H. Hoos, and Kevin Leyton-Brown. Sequential model-based optimization for general algorithm configuration. In *LION*, 2011.

[27] Andrew Ilyas, Logan Engstrom, Anish Athalye, and Jessy Lin. Black-box adversarial attacks with limited queries and information. In *International conference on machine learning*, pages 2137–2146. PMLR, 2018.

[28] Kevin G. Jamieson, Robert D. Nowak, and Benjamin Recht. Query complexity of derivative-free optimization. In *NeurIPS*, pages 2681–2689, 2012.

[29] Kaiyi Ji, Zhe Wang, Yi Zhou, and Yingbin Liang. Improved zeroth-order variance reduced algorithms and analysis for nonconvex optimization. In *ICML*, pages 3100–3109. PMLR, 2019.

[30] David Kozak, Cesare Molinari, Lorenzo Rosasco, Luis Tenorio, and Silvia Villa. Zeroth-order optimization with orthogonal random directions. *Math. Program.*, 199(1):1179–1219, 2023.

[31] Liam Li and Ameet Talwalkar. Random search and reproducibility for neural architecture search. In *Proceedings of the Thirty-Fifth Conference on Uncertainty in Artificial Intelligence, UAI 2019, Tel Aviv, Israel, July 22-25, 2019*, volume 115 of *Proceedings of Machine Learning Research*, pages 367–377. AUAI Press, 2019.

[32] Hanxiao Liu, Karen Simonyan, and Yiming Yang. DARTS: differentiable architecture search. In *7th International Conference on Learning Representations, ICLR 2019, New Orleans, LA, USA, May 6-9, 2019*. OpenReview.net, 2019.

[33] Sijia Liu, Jie Chen, Pin-Yu Chen, and Alfred Hero. Zeroth-order online alternating direction method of multipliers: Convergence analysis and applications. In *International Conference on Artificial Intelligence and Statistics*, pages 288–297. PMLR, 2018.

[34] Sijia Liu, Pin-Yu Chen, Xiangyi Chen, and Mingyi Hong. signsgd via zeroth-order oracle. In *International conference on learning representations*. International Conference on Learning Representations, ICLR, 2019.

[35] Sijia Liu, Pin-Yu Chen, Bhavya Kailkhura, Gaoyuan Zhang, Alfred O. Hero III, and Pramod K. Varshney. A primer on zeroth-order optimization in signal processing and machine learning: Principals, recent advances, and applications. *IEEE Signal Process. Mag.*, 37(5):43–54, 2020.

[36] Alvaro Maggiar, Andreas Wachter, Irina S Dolinskaya, and Jeremy Staum. A derivative-free trust-region algorithm for the optimization of functions smoothed via gaussian convolution using adaptive multiple importance sampling. *SIAM Journal on Optimization*, 28(2):1478–1507, 2018.

[37] Dhruv Malik, Ashwin Pananjady, Kush Bhatia, Koulik Khamaru, Peter L. Bartlett, and Martin J. Wainwright. Derivative-free methods for policy optimization: Guarantees for linear quadratic systems. *J. Mach. Learn. Res.*, 21:21:1–21:51, 2020.

[38] Arkadij Semenovič Nemirovskij and David Borisovich Yudin. Problem complexity and method efficiency in optimization. *SIAM J. OPTIM*, 1983.

[39] Yurii Nesterov et al. Random gradient-free minimization of convex functions. Technical report, Université catholique de Louvain, Center for Operations Research and Econometrics (CORE), 2011.

[40] Hieu Pham, Melody Guan, Barret Zoph, Quoc Le, and Jeff Dean. Efficient neural architecture search via parameters sharing. In *ICML*, pages 4095–4104. PMLR, 2018.

[41] Luis Miguel Rios and Nikolaos V. Sahinidis. Derivative-free optimization: a review of algorithms and comparison of software implementations. *J. Glob. Optim.*, 56(3):1247–1293, 2013.

[42] Hao-Jun M Shi, Yuchen Xie, Richard Byrd, and Jorge Nocedal. A noise-tolerant quasi-newton algorithm for unconstrained optimization. *SIAM Journal on Optimization*, 32(1):29–55, 2022.

[43] Jasper Snoek, Hugo Larochelle, and Ryan P. Adams. Practical bayesian optimization of machine learning algorithms. In *NeurIPS*, 2012.

[44] Yun-Yun Tsai, Pin-Yu Chen, and Tsung-Yi Ho. Transfer learning without knowing: Reprogramming black-box machine learning models with scarce data and limited resources. In *International Conference on Machine Learning*, pages 9614–9624. PMLR, 2020.

[45] Anirudh Vemula, Wen Sun, and J. Andrew Bagnell. Contrasting exploration in parameter and action space: A zeroth-order optimization perspective. In *AISTATS*, volume 89 of *Proceedings of Machine Learning Research*, pages 2926–2935. PMLR, 2019.

[46] Xiaoxing Wang, Xiangxiang Chu, Yuda Fan, Zhexi Zhang, Bo Zhang, Xiaokang Yang, and Junchi Yan. ROME: robustifying memory-efficient NAS via topology disentanglement and gradient accumulation. In *ICCV*, pages 5916–5926. IEEE, 2023.

[47] Xiaoxing Wang, Wenxuan Guo, Jianlin Su, Xiaokang Yang, and Junchi Yan. Zarts: On zero-order optimization for neural architecture search. *Advances in Neural Information Processing Systems*, 35:12868–12880, 2022.

[48] Xiaoxing Wang, Jiaxing Li, Chao Xue, Wei Liu, Weifeng Liu, Xiaokang Yang, Junchi Yan, and Dacheng Tao. Poisson process for bayesian optimization. In *AutoML*, volume 224 of *Proceedings of Machine Learning Research*, pages 3/1–20, 2023.

[49] Xiaoxing Wang, Zhirui Lian, Jiale Lin, Chao Xue, and Junchi Yan. DIY your easynas for vision: Convolution operation merging, map channel reducing, and search space to supernet conversion tooling. *IEEE Trans. Pattern Anal. Mach. Intell.*, 45(11):13974–13990, 2023.

[50] Xiaoxing Wang, Jiale Lin, Juanping Zhao, Xiaokang Yang, and Junchi Yan. Eautodet: Efficient architecture search for object detection. In *ECCV*, volume 13680 of *Lecture Notes in Computer Science*, pages 668–684, 2022.

[51] Xiaoxing Wang, Chao Xue, Junchi Yan, Xiaokang Yang, Yonggang Hu, and Kewei Sun. Mergenas: Merge operations into one for differentiable architecture search. In *IJCAI*, pages 3065–3072, 2020.

[52] Yining Wang, Simon Du, Sivaraman Balakrishnan, and Aarti Singh. Stochastic zeroth-order optimization in high dimensions. In *International conference on artificial intelligence and statistics*, pages 1356–1365. PMLR, 2018.

[53] Beichen Zhang, Xiaoxing Wang, Xiaohan Qin, and Junchi Yan. Boosting order-preserving and transferability for neural architecture search: a joint architecture refined search and fine-tuning approach. In *Proceedings of the IEEE/CVF Conference on Computer Vision and Pattern Recognition*, pages 5662–5671, 2024.

[54] Yan Zhang, Yi Zhou, Kaiyi Ji, and Michael M. Zavlanos. A new one-point residual-feedback oracle for black-box learning and control. *Autom.*, 136:110006, 2022.

[55] Yihua Zhang, Pingzhi Li, Junyuan Hong, Jiaxiang Li, Yimeng Zhang, Wenqing Zheng, Pin-Yu Chen, Jason D. Lee, Wotao Yin, Mingyi Hong, Zhangyang Wang, Sijia Liu, and Tianlong Chen. Revisiting zeroth-order optimization for memory-efficient LLM fine-tuning: A benchmark. In *ICML*, 2024.

# A Convergence Analysis

## A.1 Proof of Lemma 3.1

**Lemma 3.1** Let $f$ be a function satisfying Assumption 3.1 and 3.2. For every $k \in \mathbb{N}$, $\Delta_{x_k} \in \mathbb{R}^{n \times d}$ and $\|\Delta_{x_k}^i\| < b$, we have:

$$\|\Delta_{x_k} g(x_k) - \Delta_{x_k} \nabla f(x_k)\| \leq \frac{Lb\sqrt{n}}{2}.$$

*Proof.* We start with the Taylor's expansion of function $f$ defined in Eq. 1

$$g(x)^T \Delta_x^i = f(x + \Delta_x^i) - f(x) = \int_0^1 \Delta_x^{i\top} \nabla f(x + t\Delta_x^i) dt$$

$$g(x)^\top \Delta_x^i - \nabla f(x)^\top \Delta_x^i = \int_0^1 \Delta_x^{i\top} (\nabla f(x + t\Delta_x^i) - \nabla f(x)) dt$$

$$(\textbf{Assumption 3.2}) \qquad \leq \int_0^1 \Delta_x^{i\top} \Delta_x^i L t dt$$

$$= \frac{L\|\Delta_x^i\|^2}{2},$$

thus

$$\|\Delta_x (g(x) - \nabla f(x))\| \leq \frac{L\|\Delta_x\|^2}{2}.$$

With the condition $\|\Delta_{x_k}^i\| < b$, we then derive that

$$\|\Delta_{x_k} (g(x_k) - \nabla f(x_k))\| \leq \frac{Lb\sqrt{n}}{2},$$

which completes the proof. $\qquad \square$

## A.2 Proof of Lemma 3.2

**Lemma 3.2** Let the SVD of $\Delta_{x_k} = U\Lambda V^\top$, $A_{x_k}^{-1} = (\Delta_{x_k} \Delta_{x_k}^\top)^{-1}$, $\rho_k$ denotes the spectral radius of $A_{x_k}^{-1}$. Based on Lemma 3.1, we have:

$$\|g(x_k) - \nabla f(x_k)\|^2 \leq r\|\nabla f(x_k)\|^2 + \frac{\rho_k L^2 b^2 n}{4},$$

where

$$r = \frac{\sum_{j=n}^d v_j^2}{\|v\|^2} < 1, \quad v = V^\top \nabla f(x_k).$$

*Proof.* Without losing generality, we omit the subscript $k$ in our proof. $g(x) = \Delta_x^\top A^{-1} \Delta_f$ and $\Delta_f = \Delta_x g(x) = \Delta_x[g(x) - \nabla f(x)] + \Delta_x \nabla f(x)$, we define

$$E = \Delta_x[g(x) - \nabla f(x)],$$

then we have:

$$\|g(x) - \nabla f(x)\|^2 = \|\Delta_x^\top A^{-1} \Delta_f - \nabla f(x)\|^2$$

$$= \|(\Delta_x^\top A^{-1} \Delta_x - \mathbb{I}^{d\times d}) \nabla f(x) + \Delta_x^\top A^{-1} E\|^2,$$

where $\mathbb{I}^{d\times d}$ denotes an identity matrix with $d$ rows and columns. Suppose the SVD of $\Delta_x = U\Lambda V^\top$, substitute to $\Delta_x^\top A^{-1} \Delta_x$, we have:

$$\Delta_x^\top A^{-1} \Delta_x - \mathbb{I}^{d\times d} = V\Lambda^\top (\Lambda\Lambda^\top)^{-1} \Lambda V^\top - \mathbb{I}^{d\times d}$$

$$= V \begin{bmatrix} \mathbb{I}^{n\times n} & 0 \\ 0 & 0 \end{bmatrix} V^\top - \mathbb{I}^{d\times d} = V \begin{bmatrix} 0 & 0 \\ 0 & \mathbb{I}^{(d-n)\times(d-n)} \end{bmatrix} V^\top \triangleq V\tilde{\mathbb{I}}^{d-n} V^\top.$$

Then, we can simplify $\|g(x) - \nabla f(x)\|$ as:

$$\begin{aligned}
\|g(x) - \nabla f(x)\|^2 &= \|V\tilde{\mathbb{I}}^{d-n}V^\top\nabla f(x) + \Delta_x^\top A^{-1}E\|^2 \\
&= \|V\tilde{\mathbb{I}}^{d-n}V^\top\nabla f(x)\|^2 + \|\Delta_x^\top A^{-1}E\|^2 + 2\langle V\tilde{\mathbb{I}}^{d-n}V^\top\nabla f(x), \Delta_x^\top A^{-1}E\rangle,
\end{aligned}$$

where $\langle x, y\rangle$ denotes the inner product of vectors $x$ and $y$. According to the definition of inner product and the SVD of $\Delta_x = U\Lambda V^T$, we have:

$$\begin{aligned}
\langle V\tilde{\mathbb{I}}^{d-n}V^\top\nabla f(x), \Delta_x^\top A^{-1}E\rangle &= \nabla f(x)^T V(\tilde{\mathbb{I}}^{d-n})^\top V^\top \Delta_x^\top A^{-1}E \\
&= \nabla f(x)^T V\big[(\tilde{\mathbb{I}}^{d-n})^\top \Lambda^\top (\Lambda\Lambda^\top)^{-1}\big]UE = 0.
\end{aligned}$$

Moreover, we denote $v = V^\top\nabla f(x)$, then $\|V\tilde{\mathbb{I}}^{d-n}V^\top\nabla f(x)\|^2 = \sum_{j=n}^d v_j^2$, we define $r$ as

$$r = \frac{\sum_{j=n}^d v_j^2}{\|\nabla f(x)\|^2} = \frac{\sum_{j=n}^d v_j^2}{\|v\|^2} < 1.$$

Therefore,

$$\begin{aligned}
\|g(x) - \nabla f(x)\|^2 &= r\|\nabla f(x)\|^2 + \|\Delta_x^\top A^{-1}E\|^2 \\
&= r\|\nabla f(x)\|^2 + E^\top A^{-1}E \leq r\|\nabla f(x)\|^2 + \rho\|E\|^2,
\end{aligned}$$

where $\rho$ is the spectral radius of $A^{-1}$. According to Lemma 3.1, we have:

$$\|g(x) - \nabla f(x)\|^2 \leq r\|\nabla f(x)\|^2 + \frac{\rho L^2 b^2 n}{4}.$$

$\square$

## A.3  Proof of Proposition 3.3

**Proposition 3.3** Suppose $f$ is a convex function and $\exists\, \theta \in [0, 1)$ satisfying $\rho_k \leq \frac{4(\theta^2 - r)\|\nabla f(x)\|^2}{L^2 b^2 n}$, we have:

$$f(x_k) - f(x^*) \leq \frac{C}{k(1 - \theta)^2},$$

*Proof.* Based on Lemma 3.2, we substitute Eq. 13 with $\rho_k \leq \frac{4(\theta^2 - r)\|\nabla f(x)\|^2}{L^2 b^2 n}$, then we have

$$\|g(x_k) - \nabla f(x_k)\| \leq \theta\|\nabla f(x_k)\|.$$

This condition is also referred to as *norm condition* and is introduced and studied in [9] in the context of trust-region methods with inaccurate gradients. This then yeilds

$$(1 - \theta)\|\nabla f(x_k)\| \leq \|g(x_k)\| \leq (1 + \theta)\|\nabla f(x_k)\| \tag{16}$$

by using Cauchy-Schwarz inequality. Note that our iteration also satisfies the Armijo condition

$$f(x_{k+1}) \leq f(x_k) - c_1\sigma_k\|g(x_k)\|^2, \tag{17}$$

where $c_1 \in (0, 1)$ is a constant, and $\sigma_k$ is the magnitude of descent direction found by using Armijo backtracking line search. By (16), we have

$$\begin{aligned}
f(x_{k+1}) &\leq f(x_k) - c_1\sigma_k\|g(x_k)\|^2 \\
&\leq f(x_k) - c_1\sigma_k(1 - \theta)^2\|\nabla f(x_k)\|^2.
\end{aligned}$$

**Assumption A.1. (Convexity and bounded level sets of $f$)** The function $f$ is convex and has bounded level sets, i.e.,

$$\|x - x^*\| \leq D, \quad for\ all\ x\ with\ f(x) \leq f(x_0),$$

where $x^*$ is a global minimizer of $f$. Let $f^* = f(x^*)$.

By Assumption A.1, we have

$$
\begin{aligned}
f(x_k) - f(x^*) &\leq \nabla f(x_k)^\top (x_k - x^*) \\
&\leq \|\nabla f(x_k)\| \cdot \|x_k - x^*\| \\
&\leq D \cdot \|\nabla f(x_k)\|.
\end{aligned}
$$

Let $z_k = f(x_k) - f*$, thus

$$
\begin{aligned}
z_k - z_{k+1} &= f(x_k) - f(x_{k+1}) \\
&\geq c_1 \sigma_k (1-\theta)^2 \|\nabla f(x_k)\|^2 \\
&\geq \frac{c_1 \sigma_k (1-\theta)^2 z_k^2}{D^2},
\end{aligned}
$$

which implies

$$
\frac{1}{z_{k+1}} - \frac{1}{z_k} = \frac{z_k - z_{k+1}}{z_k z_{k+1}} \geq \frac{z_k - z_{k+1}}{z_k^2} \geq \frac{c_1 \sigma_k (1-\theta)^2}{D^2}
$$

$$
\frac{1}{z_k} = \frac{1}{z_0} + \sum_{i=0}^{k-1} \left( \frac{1}{z_{i+1}} - \frac{1}{z_i} \right) \geq \frac{1}{z_0} + \frac{c_1 (1-\theta)^2 \sum_{i=0}^{k-1} \sigma_i}{D^2} \geq \frac{c_1 (1-\theta)^2 \sum_{i=0}^{k-1} \sigma_i}{D^2},
$$

thus

$$
z_k = f(x_k) - f* \leq \frac{D^2}{c_1 (1-\theta)^2 \sum_{i=0}^{k-1} \sigma_i}. \tag{18}
$$

Moreover, in line with the assumption in [30], we can set $\underline{\sigma}$ as the minimum step size in the algorithm. In fact, the previous works [6] and [42] have shown that in the convex case, the number of failed line searches can be bounded w.r.t $\underline{\sigma}$. Thus we mainly focus on the case when (17) holds. In this case, define $C = \frac{D^2}{c_1 \underline{\sigma}}$, then (18) implies

$$
z_k = f(x_k) - f* \leq \frac{D^2}{c_1 (1-\theta)^2 \sum_{i=0}^{k-1} \sigma_i} \leq \frac{D^2}{k \underline{\sigma} c_1 (1-\theta)^2} = \frac{C}{k(1-\theta)^2},
$$

which shows a sublinear convergence rate. Similar to the prior ZO works (Table 1 in [29]), we set the minimum step size $\underline{\sigma} \sim O(\frac{1}{d})$ to guarantee the correctness of the assumptions, thus the convergence rate of our method is:

$$
f(x_k) - f(x^*) \leq \frac{D^2}{k c_1 \underline{\sigma} (1-\theta)^2} = \frac{C}{k(1-\theta)^2} \sim O(\frac{d}{k}). \tag{19}
$$

$\square$

## B  Comparison of Computation Complexity

The algorithm flow is shown in Alg. 1. Suppose $n$ is the sample size, $d$ is the dimension of variables, and $n < d$. For the $k$-th iteration, suppose $n_k$ vectors are reused from the prior iterations and $n - n_k$ vectors are newly sampled. The computation complexity includes two parts: querying function evaluations for newly sampled vectors, and computing gradient estimation by Eq. 5. For the former, we denote the complexity of querying function evaluation once as $O(c(d))$, where $c(\cdot)$ could be linear, polynomial, etc.. Then querying $n - n_k$ times requires $O((n - n_k)c(d))$ computation resources. As for the latter, Eq. 5 requires to compute $A_{x_k}$ with complexity $O((n - n_k)nd)$, the inverse of matrix $A_{x_k}$ with complexity $O(n^3)$, and the matrix multiplication of $\Delta_{x_k}^\top A_{x_k}^{-1} \Delta_f$ with complexity $O(n^2 + nd)$. Since $n < d$, the overall computation complexity for the $k$-th iteration is $O(n^3 + (n - n_k)nd + (n - n_k)c(d))$. Table 5 compares the computation complexity in one iteration of our method with prior smoothing and linear interpolation methods. Except for our methods, others do not support the reusing strategy.

Table 5: Comparison of computation complexity in one iteration between various ZO methods.

| Methods | Computation Complexity |
|---|---|
| ZO-SGD [22] | $O\big(nd + nc(d)\big)$ |
| ZO-signSGD [34] | $O\big(nd + nc(d)\big)$ |
| ZO-AdaMM [12] | $O\big(nd + nc(d)\big)$ |
| Conventional Linear Interpolation (n=d) [5] | $O\big(d^3 + dc(d)\big)$ |
| Linear Interpolation w/ orthogonal vectors [30] | $O\big(n^2d + nc(d)\big)$ |
| ReLIZO (ours) | $O\big(n^3 + (n - n_k)nd + (n - n_k)c(d)\big)$ |

Table 6: Comparison of various ZO methods on small sample size $N = 2$.

| Methods | ARGTRIGLS | CHNROSNB | COATING | BOXPOWER | SROSENBR | BROYDNBDLS |
|---|---|---|---|---|---|---|
| ZO-SGD [22] | 6.03 | 376.32 | 2007.45 | 361.72 | 2706.39 | 116.9 |
| ZO-signSGD [34] | 66.33 | 224.94 | 1431.66 | 48.85 | 1398.44 | 91.39 |
| ZO-AdaMM [12] | 3.27 | 50.23 | 994.63 | 33.13 | 355.32 | 3.83 |
| ReLIZO (ours) | 2.38 | 44.38 | 831.5 | 27.44 | 115.48 | 0.65 |

Table 7: Comparison of various ZO methods on large dimension $d$.

| SROSENBR | d=1000 | d=5000 | d=10000 |
|---|---|---|---|
| ZO-SGD [22] | 293.81(277.31) | 45432.77(35091.73) | 96988.70(49867.83) |
| ZO-signSGD [34] | 883.52(867.02) | 36289.83(25948.79) | 96749.68(49628.81) |
| ZO-AdaMM [12] | 47.54(31.04) | 32071.75(21730.71) | 87781.39(41666.52) |
| ReLIZO (ours) | 16.50(0) | 10341.04(0) | 47120.87(0) |

| BOXPOWER | d=1000 | d=10000 | d=20000 |
|---|---|---|---|
| ZO-SGD [22] | 11.10(2.11) | 118.05(28.13) | 248.02(67.03) |
| ZO-signSGD [34] | 37.74(28.75) | 325.92(236) | 713.91(532.92) |
| ZO-AdaMM [12] | 20.88(11.89) | 235.39(145.47) | 471.19(290.2) |
| ReLIZO (ours) | 8.99(0) | 89.92(0) | 180.99(0) |

## C  Supplementary Experiments

### C.1  Experiments on Extreme Conditions

**Experiments on small sample size $N$:** We set $N = 2$ and compare ReLIZO and other methods on six PyCUTEst optimization tasks. The results are reported in Table 6, showing that ReLIZO also surpasses other baselines. We analyze that the methods based on smoothing strategy utilize Monte Carlo sampling to estimate the expectation of various directional derivatives and a small $N$ will lead to a biased estimation as shown in [5]. In contrast, our ReLIZO computes the projection of gradient on a space generated by arbitrary numbers of sampled directions by solving the QCLP, and thus can also work pretty well under a small $N$.

**Experiments on large dimension $d$:** We conduct experiments on two PyCUTEst optimization tasks with an adjustable variable dimension. Each optimizer updates the variables 500 times and samples 8 random directions at each iteration to update the variables. We also utilize grid search to obtain the best learning rate for each optimizer for different tasks. The best solution obtained by each optimizer are reported in Table 7, and the values in the brackets report the gap between each baseline and ReLIZO. The results show that the improvement of ReLIZO is increasing as $d$ increases for each optimization task. We analyze that the sample size $N$ becomes relatively small compared to the dimension $d$ as $d$ increases, our method computes the projection of gradient on a space generated by $N$ sampled directions by solving the QCLP, and thus can also work pretty well even if $N << d$.

## C.2 Discussion on the Line Search

We adopt line search to adaptively adjust the stepsize to ensure a fast convergence rate. Our experimental results show that there are few queries from line search during the whole optimization process in most cases. Specifically, there are 0 queries from line search in NAS task and several optimization tasks in PyCUTEst, including CHNROSNB, SROSENBR, and BOXPOWER. We report the number of queries in each task in Table 8, where '#LS' denotes the number of queries from line search, '#Queries' denotes the number of actual queries including #LS during the optimization procedure, '#Reused' denotes the number of reused queries, 'N x T' denotes the total queries required by other methods without reusing strategy ($N$ is the sample size per iteration and $T$ is the number of iterations).

Table 8: Number of queries from line search, total queries, reused samples during the optimization process.

|  | ARGTRIGLS | CHNROSNB | COATING | BOXPOWER | SROSENBR | BROYDNBDLS | NAS-Bench-201 |
|---|---|---|---|---|---|---|---|
| #LS | 115 | 0 | 77 | 0 | 0 | 90 | 0 |
| #Queries | 3689+115 | 2294+0 | 3780+77 | 2715+0 | 2526+0 | 1353+90 | 11885+0 |
| #Reused | 311 | 1706 | 220 | 1285 | 1474 | 2647 | 7665 |
| $N \times T$ | 4000 | 4000 | 4000 | 4000 | 4000 | 4000 | 19550 |

## C.3 Ablation Studies on Other Problems

Results of ablation studies on another two problems MANCINO and ARGTRIGLS are shown in Fig. 4.

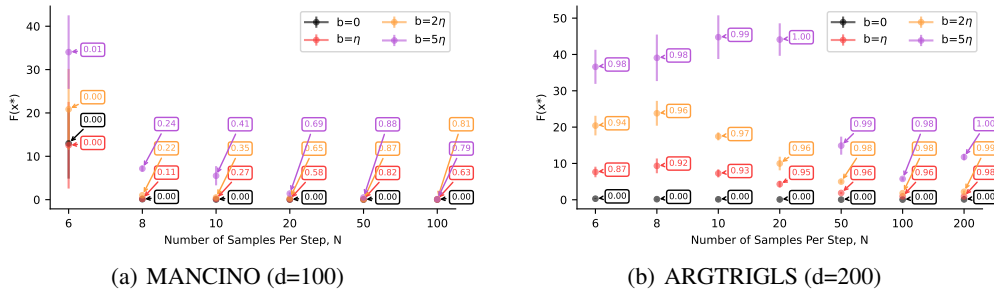

(a) MANCINO (d=100)

(b) ARGTRIGLS (d=200)

Figure 4: Illustration of the best function value found by our method in 500 iterations under different settings of sample size $N$ and reusable distance bound $b$. We conduct three replication experiments at each setting and plot the average performance and standard deviation as nodes and lines. We also illustrate the reusing rates as the floats in the boxes beside the scatters.

We also report the optimization procedure of our method at different settings of reusable distance bound $b$ in Fig. 5 and sample size $n$ in Fig. 6.

## C.4 Improved Sampling Strategy with Momentum

The ability of ReLIZO to sample new vectors from an arbitrary distribution can be considered one of its strengths compared to other ZO methods. In contrast, smoothing-based ZO methods have to rely on Gaussian or uniform distributions with a mean of zero. We introduce an effective sampling strategy inspired by SGD with momentum. Specifically, we propose a sampling momentum strategy where the sampling probability in the current iteration follows a distribution defined as

Table 9: Comparison of ReLIZO and ReLIZO-m on NAS tasks.

| Method | CIFAR10-valid | CIFAR10-test | CIFAR100-valid | CIFAR100-test | ImageNet-16-120-valid | ImageNet-16-120-test |
|---|---|---|---|---|---|---|
| ReLIZO | 89.50 | 92.45 | 69.00 | 69.03 | 42.09 | 42.31 |
| ReLIZO-m | 90.71 | 93.41 | 70.40 | 69.63 | 42.79 | 43.22 |

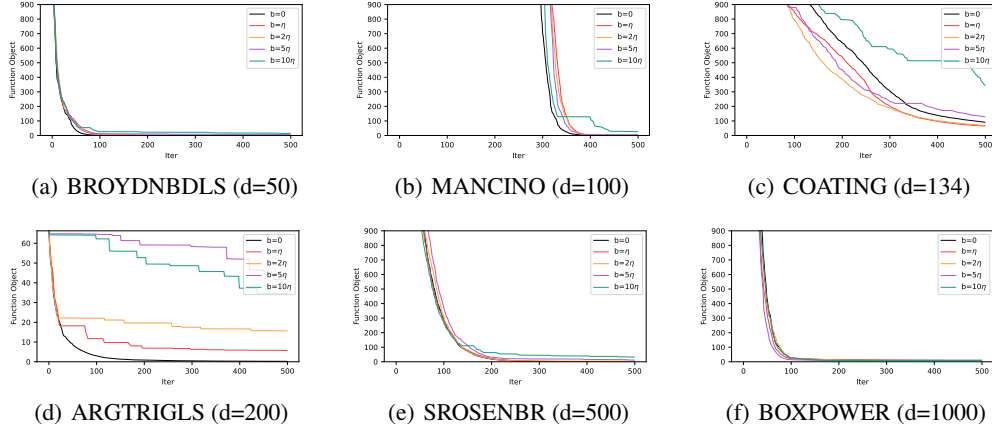

Figure 5: Illustration of the optimization procedure of our method at different settings of reusable distance bound $b$. The sample size is fixed as 8.

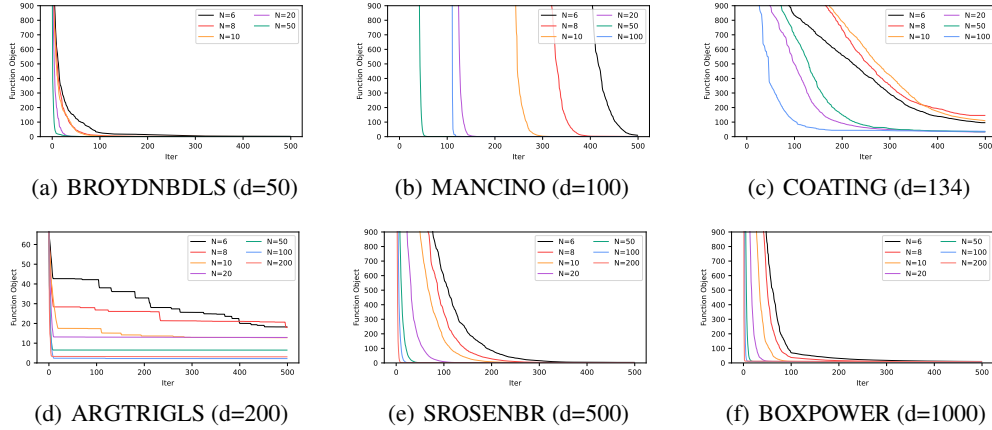

Figure 6: Illustration of the optimization procedure of our method with sample size $n$. The reusable distance bound is fixed as $2\eta$.

$p(x_t) = \alpha N(g_{t-1}, \sigma) + (1 - \alpha)N(0, \sigma)$, with $N()$ denoting a Gaussian distribution, $g_{t-1}$ is the estimated gradient in the last iteration, $\alpha$ as the momentum parameter, and $\sigma$ as the standard deviation. The results of this approach, labeled as ReLIZO-m, are presented in Table 9.

## C.5 Additional Experimental Results on the Reusable Distance Bound $b$

In Fig. 5, we observe a relative performance drop (slightly slower convergence) when the reusable distance bound $b \geq \eta$. However, this does not imply that the reuse strategy is ineffective in this case. The ARGTRIGLS problem is characterized by a rugged landscape with numerous local minima. In this case, as shown in Fig. 4(b), even with $b = \eta$ and a sample size $n = 8$, the reuse rate of ARGTRIGLS exceeds 90%, whereas reuse rates for other problems are generally below 50% (e.g. MANCINO with d=100 in Fig. 4(a), SROSENBR with d=500 in Fig. 2(b)). This high reuse rate indicates that the number of new queries during optimization is relatively small, contributing to the observed decrease in convergence speed. To address this issue, we can employ simple strategies such as reducing the reusable distance bound $b$. Table 10 illustrates that performance improves as $b$ decreases.

Table 10: The objective value and total reuse rate of different reusable distance bound $b$.

| reusable distance bound | $b = 0$ | $b = 0.01\eta$ | $b = 0.05\eta$ | $b = 0.1\eta$ | $b = 0.5\eta$ | $b = \eta$ |
|---|---|---|---|---|---|---|
| objective value (iter=500) | 0.133 | 0.162 | 0.298 | 0.495 | 3.768 | 8.158 |
| objective value (iter=2000) | 0.053 | 0.063 | 0.166 | 0.311 | 1.834 | 3.825 |
| total reuse rate | 0% | 16.6% | 72.4% | 82.3% | 94.7% | 96.8% |

Moreover, setting an upper bound for the reuse rate in each iteration can also help. Table 11 reports the performance when the maximum reuse rate in each iteration is restricted to 50%. We observe that with $b \leq 0.05\eta$ and 2000 iterations, there is minimal performance degradation, even with a reuse rate of approximately 30%.

Table 11: The objective value of different reusable distance bound $b$ with a restricted maximum reuse rate.

| reusable distance bound | $b = 0$ | $b = 0.01\eta$ | $b = 0.05\eta$ | $b = 0.1\eta$ | $b = 0.5\eta$ | $b = \eta$ |
|---|---|---|---|---|---|---|
| objective value (iter=500) | 0.133 | 0.164 | 0.183 | 0.255 | 0.312 | 0.341 |
| objective value (iter=2000) | 0.053 | 0.046 | 0.054 | 0.063 | 0.075 | 0.074 |
| total reuse rate | 0% | 11.4% | 34.2% | 41.9% | 47.9% | 48.9% |

